# Neural network computation by
# *in vitro* transcriptional circuits

**Jongmin Kim**[1]**, John J. Hopfield**[3]**, Erik Winfree**[2]
Biology[1], CNS and Computer Science[2], California Institute of Technology.
Molecular Biology[3], Princeton University.
{jongmin,winfree}@dna.caltech.edu, hopfield@princeton.edu

## Abstract

The structural similarity of neural networks and genetic regulatory networks to digital circuits, and hence to each other, was noted from the very beginning of their study [1, 2]. In this work, we propose a simple biochemical system whose architecture mimics that of genetic regulation and whose components allow for *in vitro* implementation of arbitrary circuits. We use only two enzymes in addition to DNA and RNA molecules: RNA polymerase (RNAP) and ribonuclease (RNase). We develop a rate equation for *in vitro* transcriptional networks, and derive a correspondence with general neural network rate equations [3]. As proof-of-principle demonstrations, an associative memory task and a feedforward network computation are shown by simulation. A difference between the neural network and biochemical models is also highlighted: global coupling of rate equations through enzyme saturation can lead to global feedback regulation, thus allowing a simple network without explicit mutual inhibition to perform the winner-take-all computation. Thus, the full complexity of the cell is not necessary for biochemical computation: a wide range of functional behaviors can be achieved with a small set of biochemical components.

## 1 Introduction

Biological organisms possess an enormous repertoire of genetic responses to everchanging combinations of cellular and environmental signals. Characterizing and decoding the connectivity of the genetic regulatory networks that govern these responses is a major challenge of the post-genome era [4]. Understanding the operation of biological networks is intricately intertwined with the ability to create sophisticated biochemical networks *de novo*. Recent work developing synthetic genetic regulatory networks has focused on engineered circuits in bacteria wherein protein signals are produced and degraded [5, 6]. Although remarkable, such network implementations in bacteria have many unknown and uncontrollable parameters.

We propose a biochemical model system – a simplified analog of genetic regulatory circuits – that provides well-defined connectivity and uses nucleic acid species as fuel and signals that control the network. Our goal is to establish an explicit model to guide the laboratory construction of synthetic biomolecular systems in which every component is known and

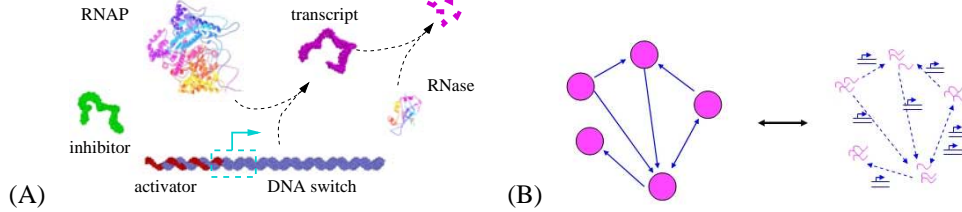

Figure 1: **(A)** The components of an *in vitro* circuit. The switch template (blue) is shown with the activator (red) attached. The dotted box indicates the promoter sequence and the downstream direction. **(B)** The correspondence between a neural network and an *in vitro* biochemical network. Neuron activity corresponds to RNA transcript concentration, while synaptic connections correspond to DNA switches with specified input and output.

where quantitative predictions can be tested. Only two enzymes are used in addition to synthetic DNA templates: RNA polymerase, which recognizes a specific promoter sequence in double-stranded DNA and transcribes the downstream DNA to produce an RNA transcript, and ribonuclease, which degrades RNA but not DNA. In this system, RNA transcript concentrations are taken as signals. Synthetic DNA templates may assume two different conformations with different transcription efficiency: ON or OFF. Upon interaction with a RNA transcript of the appropriate sequence, the DNA template switches between different conformations like a gene regulated by transcription factors. The connectivity – which RNA transcripts regulate which DNA templates – is dictated by Watson–Crick base-pairing rules and is easy to program. The network computation is powered by rNTP that drives the synthesis of RNA signals by RNAP, while RNase forces transient signals to decay. With a few assumptions, we find that this stripped-down analog of genetic regulatory networks is mathematically equivalent to recurrent neural networks, confirming that a wide range of programmable dynamical behaviors is attainable.

## 2   Construction of the transcriptional network

**The DNA transcriptional switch.** The elementary unit of our networks will be a DNA switch, which serves the role of a gene in a genetic regulatory circuit. The basic requirements for a DNA switch are to have separate input and output domains, to transcribe poorly by itself [7], and to transcribe efficiently when an activator is bound to it. A possible mechanism of activation is the complementation of an incomplete promoter region, allowing more favorable binding of RNAP to the DNA template. Figure 1A illustrates our proposed design for DNA transcriptional switches and circuits. We model a single DNA switch with the following binding reactions:

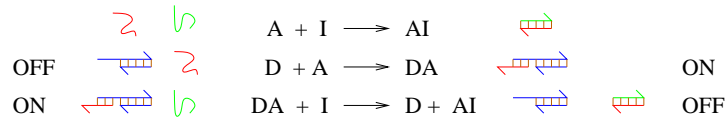

$$
\begin{aligned}
&\text{OFF} && A + I \longrightarrow AI && \text{ON}\\
&\text{ON} && D + A \longrightarrow DA && \text{OFF}\\
& && DA + I \longrightarrow D + AI &&
\end{aligned}
$$

where $D$ (blue) is a DNA template with an incomplete promoter region, $A$ (red) is an activator that complements the incomplete promoter region, and $I$ (green) is an inhibitor complementary to $A$. Thus, $I$ can bind free $A$. Furthermore, activator $A$ contains a "toehold" region [8] that overhangs past the end of $D$, allowing inhibitor $I$ to strip off $A$ from the $DA$ complex. $D$ is considered OFF and $DA$ is considered ON, based on their efficiency as templates for transcription. This set of binding reactions provides a means to choose the threshold of the sigmoidal activation function, as will be explained later.

RNAP and RNase drive changes in RNA transcript concentration; their activity is modeled using a first-order approximation for enzyme kinetics. For the moment, we assume that the input species (activator and inhibitor) are held at constant levels by external control.

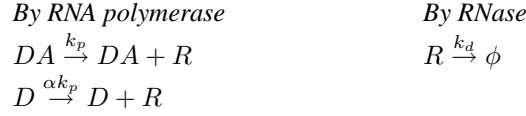

$$\textit{By RNA polymerase}$$
$$DA \xrightarrow{k_p} DA + R$$
$$D \xrightarrow{\alpha k_p} D + R$$

$$\textit{By RNase}$$
$$R \xrightarrow{k_d} \phi$$

where $0 < \alpha < 1$ due to lack of activation and $\phi$ represents the complete degradation of RNA products by RNase. $k_d$ and $k_p$ are set by the concentration of enzymes.

In general, a set of chemical reactions obeying mass action have dynamics described by

$$\frac{d[X_i]}{dt} = \sum_{\beta} k_{\beta} \prod_{j} [X_j]^{r_j^{\beta}} (p_i^{\beta} - r_i^{\beta})$$

where $k_{\beta}$ is the rate constant, $r_i^{\beta}$ is the stoichiometry of species $X_i$ as a reactant (typically 0 or 1), and $p_i^{\beta}$ is the stoichiometry of $X_i$ as a product in reaction $\beta$. Analysis of our system is greatly simplified by the assumption that the binding reactions are fast and go to completion. We define $D^{tot}$ as the sum of free and bound species: $D^{tot} = [D] + [DA]$. Similarly, $I^{tot} = [I] + [AI]$ and $A^{tot} = [A] + [DA] + [AI]$. Then, $[DA]$ depends on $D^{tot}$ and $\Delta$, where $\Delta = A^{tot} - I^{tot}$. Because $I$ can scavenge $A$ whether the latter is free or bound to $D$, $A$ can activate $D$ only when $\Delta > 0$. The amount of $[DA]$ is proportional to $\Delta$ when $0 < \Delta < D^{tot}$, as shown in Figure 2A. It is convenient to represent this nonlinearity using a piecewise-linear approximation of a sigmoidal function, specifically, $\sigma(x) = \frac{|x+1| - |x-1|}{2}$. Thus, we can represent $[DA]$ using $\sigma$ and a rescaled $\Delta$: $[DA] = \frac{1}{2} D^{tot}(1 + \sigma(\hat{\Delta}))$, where $\hat{\Delta} = \frac{2\Delta}{D^{tot}} - 1$ is called the signal activity. At steady-state, $k_d[R] = k_p[DA] + \alpha k_p[D]$; thus,

$$[R] = \frac{1}{2} \frac{k_p}{k_d} D^{tot}((1 - \alpha)\sigma(\hat{\Delta}) + 1 + \alpha) .$$

If we consider the activator concentration as an input and the steady-state transcript concentration as an output, then the (presumed constant) inhibitor concentration, $I^{tot}$, sets the threshold, and the function assumes a sigmoidal shape (Fig. 2D). Adjusting the amount of template, $D^{tot}$, sets the magnitude of the output signal and the width of the transition region (Fig. 2C). We can adjust the width of the transition region independent of the threshold such that a step function would be achieved in the limit. Thus, we have a sigmoidal function with an adjustable threshold, without reliance on cooperative binding of transcription factors as is common in biological systems [9].

**Networks of transcriptional switches.** The input domain of a DNA switch is upstream of the promoter region; the output domain is downstream of the promoter region. This separation of domains allows us to design DNA switches that have any desired connectivity.

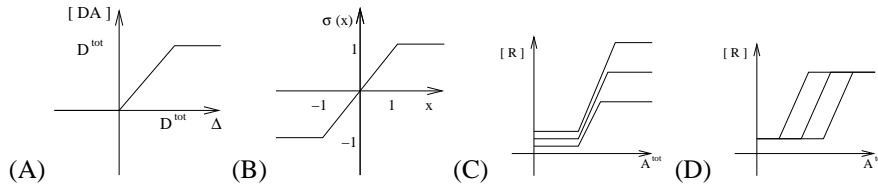

Figure 2: **(A)** $[DA]$ as a function of $\Delta$. **(B)** The sigmoid $\sigma(x)$. **(C,D)** $[R]$ as a function of $A^{tot}$ for three values of $D^{tot}$ and $I^{tot}$, respectively.

We assume that distinct signals in the network are represented as distinct RNA sequences that have negligible crosstalk (undesired binding of two molecules representing different signals). The set of legitimate binding reactions is as follows:

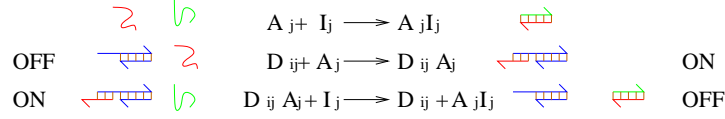

$$
\begin{aligned}
A_j + I_j &\longrightarrow A_j I_j \\
D_{ij} + A_j &\longrightarrow D_{ij} A_j \\
D_{ij} A_j + I_j &\longrightarrow D_{ij} + A_j I_j
\end{aligned}
$$

OFF     ON

ON     OFF

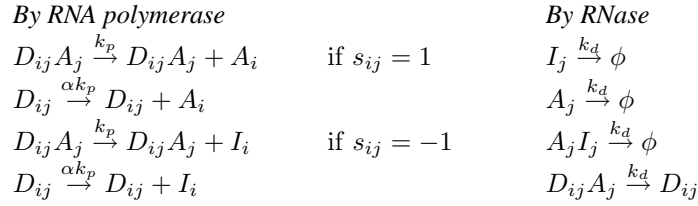

where $D_{ij}$ is the DNA template that has the $j$th input domain and $i$th output domain, the activator $A_j$ complements the incomplete promoter region of $D_{ij}$, and the inhibitor $I_j$ is complementary to $A_j$. Note that $I_j$ can strip off $A_j$ from the $D_{ij}A_j$ complex, thus imposing a sharp threshold as before. Again, we assume fast and complete binding reactions.

The set of enzyme reactions for the transcriptional network is as follows:

| *By RNA polymerase* | | *By RNase* |
|---|---|---|
| $D_{ij}A_j \xrightarrow{k_p} D_{ij}A_j + A_i$ | if $s_{ij} = 1$ | $I_j \xrightarrow{k_d} \phi$ |
| $D_{ij} \xrightarrow{\alpha k_p} D_{ij} + A_i$ | | $A_j \xrightarrow{k_d} \phi$ |
| $D_{ij}A_j \xrightarrow{k_p} D_{ij}A_j + I_i$ | if $s_{ij} = -1$ | $A_j I_j \xrightarrow{k_d} \phi$ |
| $D_{ij} \xrightarrow{\alpha k_p} D_{ij} + I_i$ | | $D_{ij}A_j \xrightarrow{k_d} D_{ij}$ |

where $s_{ij} \in \{+1, -1\}$ indicates whether switch $ij$ will produce an activator or an inhibitor. This notation reflects that the production of $I_i$ is equivalent to the consumption of $A_i$. The change of RNA concentrations over time is easy to express with $\Delta_i = A_i^{tot} - I_i^{tot}$:

$$
\frac{d\Delta_i}{dt} = -k_d \cdot \Delta_i + k_p \sum_j s_{ij}([D_{ij}A_j] + \alpha[D_{ij}]) . \tag{1}
$$

**Network equivalence.** We show next that the time evolution of this biochemical network model is equivalent to that of a general Hopfield neural network model [3]:

$$
\tau \frac{dx_i}{dt} = -x_i + \sum_j w_{ij}\sigma(x_j) + \theta_i . \tag{2}
$$

Equation 1 can be rewritten to use the same nonlinear activation function $\sigma$ defined earlier. Let $\hat{\Delta}_i = \frac{2\Delta_i}{D_{*i}^{tot}} - 1$ be a rescaled difference between activator and inhibitor concentrations, where $D_{*i}^{tot}$ is the load on $A_i$, i.e., the total concentration of all switches that bind to $A_i$: $D_{*i}^{tot} = \sum_j D_{ji}^{tot}$ and $D_{ij}^{tot} = [D_{ij}A_j] + [D_{ij}]$. Then, we can derive the following rate equation, where $\hat{\Delta}_i$ plays the role of unit $i$'s activity $x_i$:

$$
\frac{1}{k_d}\frac{d\hat{\Delta}_i}{dt} = -\hat{\Delta}_i + \sum_j \left( \frac{k_p}{k_d}(1-\alpha)s_{ij}\frac{D_{ij}^{tot}}{D_{*i}^{tot}} \right)\sigma(\hat{\Delta}_j) + \left( \sum_j \frac{k_p}{k_d}(1+\alpha)s_{ij}\frac{D_{ij}^{tot}}{D_{*i}^{tot}} - 1 \right) . \tag{3}
$$

Given the set of constants describing an arbitrary transcriptional network, the constants for an equivalent neural network can be obtained immediately by comparing Equations 2 and 3. The time constant $\tau$ is the inverse of the RNase degradation rate: fast turnover of RNA molecules leads to fast response of the network. The synaptic weight $w_{ij}$ is proportional to the concentration of switch template $ij$, attenuated by the load on $A_i$. However, the threshold $\theta_i$ is dependent on the weights, perhaps implying a lack of generality. To implement an arbitrary neural network, we must introduce two new types of switches to the transcriptional network. To achieve arbitrary thresholds, we introduce bias switches $D_{iB}$ which

have no input domain and thus produce outputs constitutively; this adds an adjustable constant to the right hand side of Equation 3. To balance the load on $A_i$, we add null switches $D_{0i}$ which bind to $A_i$ but have no output domain; this allows us to ensure that all $D_{*i}^{tot}$ are equal. Consequently, given any neural network with weights $w_{ij}$ and thresholds $\theta_i$, we can specify concentrations $D_{ij}^{tot}$ such that the biochemical network has identical dynamics, for some $\tau$.

**Michaelis–Menten enzyme reactions.** Next, we explore the validity of our assumption that enzyme kinetics are first-order reactions. A basic but more realistic model is the Michaelis–Menten mechanism [10], in which the enzyme and substrate bind to form an enzyme-substrate complex. For example, if $E$ is RNAP,

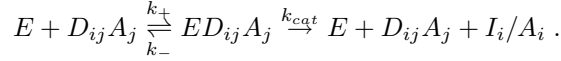

$$E + D_{ij}A_j \underset{k_-}{\overset{k_+}{\rightleftharpoons}} ED_{ij}A_j \overset{k_{cat}}{\rightarrow} E + D_{ij}A_j + I_i/A_i \ .$$

An important ramification of Michaelis–Menten reactions is that there is competition for the enzyme by the substrates, because the concentration of available enzymes is reduced as they bind to substrates, leading to saturation when the enzyme concentration is limiting. Using the steady-state assumption for Michaelis–Menten reactions, we establish the following relations to the rate constants of first-order reactions:

$$k_p = \frac{E^{tot}}{1+L} \cdot \frac{k_{cat}}{K_M} \qquad \alpha \cdot k_p = \frac{E^{tot}}{1+L} \cdot \frac{k'_{cat}}{K'_M} \qquad k_d = \frac{E_d^{tot}}{1+L_d} \cdot \frac{k_{d,cat}}{K_{d,M}} \quad (4)$$

where $k_{cat}$ and $K_M = (k_- + k_{cat})/k_+$ are the catalytic constant (enzyme's speed) and Michaelis constant (enzyme's affinity to target) of RNAP for the ON state switch, $k'_{cat}$ and $K'_M$ are for the OFF state switch, and $k_{d,cat}$ and $K_{d,M}$ are the constants of RNase. $E^{tot}$ and $E_d^{tot}$ are the concentrations of RNAP and RNase, respectively. $L = \sum_{i,j} \frac{[D_{ij}A_j]}{K_M} + \sum_{i,j} \frac{[D_{ij}]}{K'_M}$ is the load on RNAP and $L_d = \sum_{i,j} \frac{[A_j]+[I_j]+[A_jI_j]+[D_{ij}A_j]}{K_{d,M}}$ is the load on RNase (i.e., the total concentration of binding targets divided by the Michaelis constants of the enzymes), both of which may be time varying. To make the first-order approximation valid, we must keep $L$ and $L_d$ constant. Introduction of a new type of switch with different Michaelis constants can make $L$ constant by balancing the load on the enzyme. A scheme to keep $L_d$ constant is not obvious, so we set reaction conditions such that $L_d \ll 1$.

## 3 Example computations by transcriptional networks

**Feed-forward networks.** We first consider a feed-forward network to compute $f(x, y, z) = \bar{x}yz + \bar{y}z + x$. From the Boolean circuit shown in Figure 3A, we can construct an equivalent neural network. We label units 1 through 6: units 1, 2, 3 correspond to inputs $x$, $y$, $z$ whereas units 4, 5, 6 are computation units. Using the conversion rule discussed in the network equivalence section, we can calculate the parameters of the transcriptional network. Under the first-order approximation of Equation 3, the simulation result is exact (Fig. 3C). For comparison, we also explicitly simulated mass action dynamics for the full set of chemical equations with the Michaelis–Menten enzyme reactions, using biologically plausible rate constants and with $E^{tot}$ and $E_d^{tot}$ calculated from Equation 4 using estimated values of $L$ and $L_d$. The full model performs the correct calculation of $f$ for all eight 3-bit inputs, although the magnitude of signals is exaggerated due to an underestimate of RNase load (Fig. 3C).

**Associative memories.** Figure 4A shows three 4-by-4 patterns to be memorized in a continuous neural network [3]. We chose orthogonal patterns because a 16 neuron network has limited capacity. Our training algorithm is gradient descent combined with the perceptron learning rule. After training, the parameters of the neural network are converted to the parameters of the transcriptional network as previously described. Starting from a random

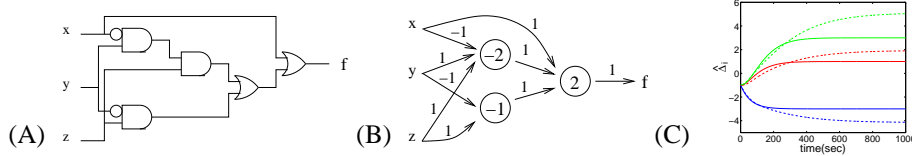

Figure 3: **(A,B)** A Boolean circuit and a neural network to compute $f(x, y, z) = \bar{x}yz + \bar{y}z + x$. **(C)** The activity of computation units (first-order approximation: solid lines; Michaelis-Menten reaction: dotted lines) for $x$=True=1, $y$=False=$-1$, $z$=True=1.

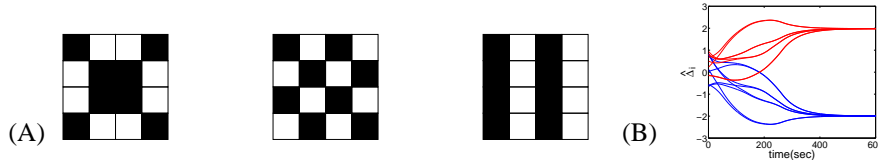

Figure 4: **(A)** The three patterns to be memorized. **(B)** Time-course for the transcriptional network recovery of the third pattern. (odd columns: blue lines, even columns: red lines)

initial state, a typical response of the transcriptional network (with the first-order approximation of Equation 3) is shown in Figure 4B. Thus, our *in vitro* transcriptional networks can support complex sets of stable steady-states.

**A winner-take-all network.** Instead of trying to compensate for the saturation phenomena of Michaelis–Menten reactions, we can make use of it for computation. As an example, consider the winner-take-all computation [11], which is commonly implemented as a neural network with $O(N^2)$ mutually inhibitory connections (Fig. 5A), but which can also be implemented as an electrical circuit with $O(N)$ interconnections by using a single global inhibitory feedback gate [12]. In a biochemical system, a limited global resource, such as RNAP, can act to regulate all the DNA switches and thus similarly produce global inhibition. This effect is exploited by the simple transcriptional network shown in Figure 5B, in which the output from each DNA switch activates the same DNA switch itself, and mutual inhibition is achieved by competition for RNAP. Specifically, we have switch templates $D_{ii}$ with fixed thresholds set by $I_i$, and $D_{ii}$ produces $A_i$ as its output RNA. With the instant binding assumption, we then derive the following equation:

$$\frac{dA_i^{tot}}{dt} = -\frac{E_d^{tot}}{1 + L_d} \cdot \frac{k_{d,cat}}{K_{d,M}} A_i^{tot} + \frac{E^{tot}}{1 + L} \left( \frac{k_{cat}}{K_M}[D_{ii}A_i] + \frac{k'_{cat}}{K'_M}[D_{ii}] \right) . \quad (5)$$

The production rate of $A_i$ depends on $A_i^{tot}$ and on $L$, while the degradation rate of $A_i$ depends on $A_i^{tot}$ and on $L_d$, as shown in Figure 6A. For a winner-take-all network, an ON state switch draws more RNAP than an OFF state switch (because of the smaller Michaelis constant for the ON state). Thus, if the other switches are turned OFF, the load on RNAP ($L$) becomes small, leading to faster production of the remaining ON switches. When the production rate curve and the degradation rate curve have three intersections, bistability is achieved such that the switches remain ON or OFF, depending on their current state.

Consider $n$ equivalent switches starting with initial activator concentrations above the threshold, and with the highest concentration at least $\delta$ above the rest (as a percentage). Analysis indicates that a less leaky system (small $\alpha$) and sufficient differences in initial activator concentrations (large $\delta$) can guarantee the existence of a unique winner. Simulations of a 10-switch winner-take-all network confirm this analysis, although we do not see perfect behavior (Fig. 6B). Figure 6C shows a time-course of a unique winner situation. Switches get turned OFF one by one whenever the activator level approaches the threshold, until only one switch remains ON.

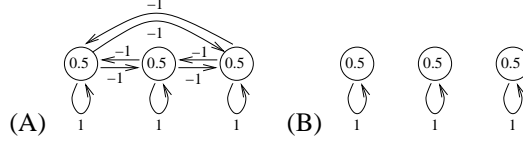

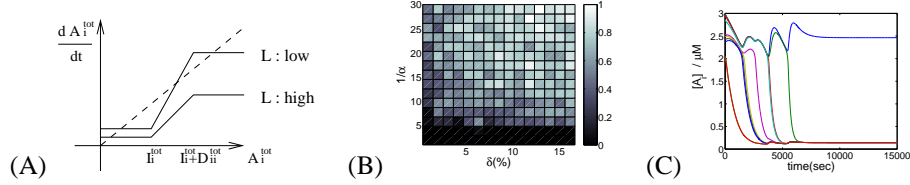

Figure 5: **(A)** A 3-unit WTA network with explicit mutual inhibition. **(B)** An equivalent biochemical network.

Figure 6: For WTA networks: **(A)** Production rates (solid lines) for two different $L$'s, compared to a linear degradation rate (dotted line). **(B)** Empirical probability of correct output as a function of $\alpha$ and $\delta$. **(C)** Time-course with $\delta = 0.33\%$ and $\alpha = 0.04$.

Similarly, we can consider a $k$-WTA network where $k$ winners persist. If we set the parameters appropriately such that $k$ winners are stable but $k + 1$ winners are unstable, the simulation result recovered $k$ winners most of the time. Even a single $k$-WTA gate can provide impressive computational power [13].

## 4  Discussion

We have shown that, if we treat transcriptionally controlled DNA switches as synapses and the concentrations of RNA species as the states of neurons, then the *in vitro* transcriptional circuit is equivalent to the neural network model and therefore can be programmed to carry out a wide variety of tasks. The structure of our biochemical networks differs from that of previous formal models of genetic regulatory circuits [14, 15, 16]. For example, consider the work of [16], which established a connection to the class of Boltzmann machines. There, the occupancy of regulatory binding sites corresponds to the state of neurons, the weights are set by the cooperative interaction among transcription factors, and the thresholds are the effective dissociation constants at a binding site. Thus, implementing a general $N$-unit neural network requires only $O(N)$ biochemical species, but up to $O(N^2)$ significant binding interactions must be encoded in the molecular sequences. Changing or tuning a network is therefore non-trivial. In contrast, in our transcriptional networks, each weight and threshold is represented by the continuously adjustable concentration of a distinct species, and the introduction or deletion of any node is straightforward.

Each synapse is represented by a DNA switch with a single input–output specification, so the number of DNA switches grows as $O(N^2)$ for a fully recurrent neural network with $N$ neurons (unlike the circuits of [16]). This constraint may be relieved because, in many networks of interest, most nodes have a small number of connections [17, 18]. The time for computation will increase as $O(N)$ due to finite hybridization rates because, if the total concentration of all RNA signals is capped, the concentration of any given species will decrease as $1/N$. The weights are capped by the maximum gain of the system, which is the production rate divided by the degradation rate. Since the time constant of the network is the inverse of the degradation rate, if we wish to implement a network with large weights, we must increase the time constant.

We can analyze the cost of computing by considering basic physical chemistry. The energy consumption is about $20kT (= 10^{-19} J)$ per nucleotide incorporated, and 1 bit of informa-

tion is encoded by a sequence containing tens of nucleotides. The encoding energy is large, since the molecule for each bit must contain specific instructions for connectivity, unlike spatially arranged digital circuits where a uniform physical signal carrier can be used. Furthermore, many copies (e.g., $10^{13}$ for a $1\mu M$ signal in $20\mu l$) of a given species must be produced to change the concentration in a bulk sample. Worse yet, because degradation is not modulated in the transcriptional network, switching relies on selective change of production rates, thus continually using energy to maintain an ON state. Devising a scheme to minimize maintenance energy costs, such as in CMOS technology for electrical circuits, is an important problem.

The theory presented here is meant to serve as a guide for the construction of real biochemical computing networks. Naturally, real systems will deviate considerably from the idealized model (although perhaps less so than do neural network models from real neurons). For example, hybridization is neither instantaneous nor irreversible, strands can have undesired conformations and crosstalk, and enzyme reactions depend on the sequence and are subject to side reactions that generate incomplete products. Some problems, such as hybridization speed and crosstalk, can be reduced by slowing the enzyme reactions and using proper sequence design [19]. Ultimately, some form of fault tolerance will be necessary at the circuit level. Restoration of outputs to digital values, achieved by any sufficiently high-gain sigmoidal activation function, provides some level of immunity to noise at the gate level, and attractor dynamics can provide restoration at the network level. A full understanding of fault tolerance in biochemical computing remains an important open question.

Future directions include utilizing the versatility of active RNA molecules (such as aptamers, ribozymes, and riboswitches [20, 21]) for more general chemical input and output, devising a biochemical learning scheme analogous to neural network training algorithms [22], and studying the stochastic behavior of the transcriptional network when a very small number of molecules are involved in small volumes [5].

**Acknowledgements.** We thank Michael Elowitz, Paul Rothemund, Casimir Wierzynski, Dan Stick and David Zhang for valuable discussions, and ONR and NSF for funding.

# References

[1] McCulloch WS, Pitts W, Bull. Math. Biophys. **5** (1943), 115.
[2] Monod J, Jacob F, Cold Spring Harb. Symp. Quant. Biol. **26** (1961), 389-401.
[3] Hopfield JJ, Proc. Nat. Acad. Sci. USA **81** (1984), 3088-3092.
[4] Hasty J, McMillen D, Issacs F, Collins JJ, Nat. Rev. Genet. **2** (2001), 268-279.
[5] Elowitz MB, Leibler S, Nature **403** (2000), 335-338.
[6] Gardner TS, Cantor CR, Collins JJ, Nature **403** (2000), 339-342.
[7] Martin CT, Coleman JE, Biochemistry **26** (1987), 2690-2696.
[8] Yurke B, Mills AP Jr., Genetic Programming and Evolvable Machines **4** (2003), 111-122.
[9] Shea MA, Ackers GK, J. Mol. Biol. **181** (1985), 211-230.
[10] Hammes GG, *Thermodynamics and kinetics for the biological sciences,* Wiley (2000).
[11] Yuille AL, Gieger D, in *The Handbook of Brain Theory and Neural Networks,* Arbib MA, ed., MIT Press (1995), 1056-1060.
[12] Tank DW, Hopfield JJ, IEEE Trans. on Circuits and Systems **33** (1986), 533-541.
[13] Maass W, Neural Computation **12** (2000), 2519-2535.
[14] Glass L, Kauffman SA, J. Theo. Biol. **39** (1973), 103-129.
[15] Mjolsness E, Sharp DH, Reinitz J, J. Theo. Biol. **152** (1991), 429-453.
[16] Buchler NE, Gerland U, Hwa T, Proc. Nat. Acad. Sci. USA **100** (2003), 5136-5141.
[17] Bray D, Science **301** (2003), 1864-1865.
[18] Reed RD, IEEE Trans. on Neural Networks, **4** (1993), 740-744.
[19] Dirks R, Lin M, Winfree E, Pierce NA, Nucleic Acids Research **32** (2004), 1392-1403.
[20] Lilley DM, Trends Biochem. Sci. **28** (2003), 495-501.
[21] Nudler E, Mironov AS, Trends Biochem. Sci. **29** (2004), 11-17.
[22] Mills AP Jr., Yurke B, Platzman PM, Biosystems **52** (1999), 175-180.
